# PAC Generalization Bounds for Co-training

**Sanjoy Dasgupta**
AT&T Labs–Research
*dasgupta@research.att.com*

**Michael L. Littman**
AT&T Labs–Research
*mlittman@research.att.com*

**David McAllester**
AT&T Labs–Research
*dmac@research.att.com*

## Abstract

The rule-based bootstrapping introduced by Yarowsky, and its co-training variant by Blum and Mitchell, have met with considerable empirical success. Earlier work on the theory of co-training has been only loosely related to empirically useful co-training algorithms. Here we give a new PAC-style bound on generalization error which justifies both the use of confidences — partial rules and partial labeling of the unlabeled data — and the use of an agreement-based objective function as suggested by Collins and Singer. Our bounds apply to the multiclass case, i.e., where instances are to be assigned one of $k$ labels for $k \geq 2$.

## 1 Introduction

In this paper, we study bootstrapping algorithms for learning from unlabeled data. The general idea in bootstrapping is to use some initial labeled data to build a (possibly partial) predictive labeling procedure; then use the labeling procedure to label more data; then use the newly labeled data to build a new predictive procedure and so on. This process can be iterated until a fixed point is reached or some other stopping criterion is met. Here we give PAC style bounds on generalization error which can be used to formally justify certain boostrapping algorithms.

One well-known form of bootstrapping is the EM algorithm (Dempster, Laird and Rubin, 1977). This algorithm iteratively updates model parameters by using the current model to infer (a probability distribution on) labels for the unlabeled data and then adjusting the model parameters to fit the (distribution on) filled-in labels. When the model defines a joint probability distribution over observable data and unobservable labels, each iteration of the EM algorithm can be shown to increase the probability of the observable data given the model parameters. However, EM is often subject to local minima — situations in which the filled-in data and the model parameters fit each other well but the model parameters are far from their maximum-likelihood values. Furthermore, even if EM does find the globally optimal maximum likelihood parameters, a model with a large number of parameters will over-fit the data. No PAC-style guarantee has yet been given for the generalization accuracy of the maximum likelihood model.

An alternative to EM is rule-based bootstrapping of the form used by Yarowsky (1995), in which one assigns labels to some fraction of a corpus of unlabeled data and then infers

new labeling rules using these assigned labels as training data. New labels lead to new rules which in turn lead to new labels, and so on. Unlike EM, rule-based bootstrapping typically does not attempt to fill in, or assign a distribution over, labels unless there is compelling evidence for a particular label. One intuitive motivation for this is that by avoiding training on low-confidence filled-in labels one might avoid the self-justifying local optima encountered by EM. Here we prove PAC-style generalization guarantees for rule-based bootstrapping.

Our results are based on an independence assumption introduced by Blum and Mitchell (1998) which is rather strong but is used by many successful applications. Consider, for example, a stochastic context-free grammar. If we generate a parse tree using such a grammar then the nonterminal symbol labeling a phrase separates the phrase from its context — the phrase and the context are statistically independent given the nonterminal symbol. More intuitively, in natural language the distribution of contexts into which a given phrase can be inserted is determined to some extent by the "type" of the phrase. The type includes the syntactic category but might also include semantic subclassifications, for instance, whether a noun phrase refers to a person, organization, or location. If we think of each particular occurrence of a phrase as a triple $\langle x_1, \ y, \ x_2 \rangle$, where $x_1$ is the phrase itself, $y$ is the "type" of the phrase, and $x_2$ is the context, then we expect that $x_1$ is conditionally independent of $x_2$ given $y$. The conditional independence can be made to hold precisely if we generate such triples using a stochastic context free grammar where $y$ is the syntactic category of the phrase.

Blum and Mitchell introduce *co-training* as a general term for rule-based bootstrapping in which each rule must be based entirely on $x_1$ or entirely on $x_2$. In other words, there are two distinct hypothesis classes, $\mathcal{H}_1$ which consists of functions predicting $y$ from $x_1$, and $\mathcal{H}_2$ which consists of functions predicting $y$ from $x_2$. A co-training algorithm bootstraps by alternately selecting $h_1 \in \mathcal{H}_1$ and $h_2 \in \mathcal{H}_2$. The principal assumption made by Blum and Mitchell is that $x_1$ is conditionally independent of $x_2$ given $y$. Under such circumstances, they show that, given a weak predictor in $\mathcal{H}_1$, and given an algorithm which can learn $\mathcal{H}_2$ under random misclassification noise, it is possible to learn a good predictor in $\mathcal{H}_2$. This gives some degree of justification for the co-training restriction on rule-based bootstrapping. However, it does not provide a bound on generalization error as a function of empirically measurable quantities. Furthermore, there is no apparent relationship between this PAC-learnability theorem and the iterative co-training algorithm they suggest.

Collins and Singer (1999) suggest a refinement of the co-training algorithm in which one explicitly optimizes an objective function that measures the degree of agreement between the predictions based on $x_1$ and those based on $x_2$. They describe methods for "boosting" this objective function but do not provide any formal justification for the objective function itself. Here we give a PAC-style performance guarantee in terms of this agreement rate. This guarantee formally justifies the Collins and Singer suggestion.

In this paper, we use *partial* classification rules, which either output a class label or output a special symbol $\bot$ indicating no opinion. The error of a partial rule is the probability that the rule is incorrect given that it has an opinion. We work in the co-training setting where we have a pair of partial rules $h_1$ and $h_2$ where $h_1$ (sometimes) predicts $y$ from $x_1$ and $h_2$ (sometimes) predicts $y$ from $x_2$. Each of the rules $h_1$ and $h_2$ can be "composite rules", such as decision lists, where each composite rule contains a large set of smaller rules within it. We give a bound on the generalization error of each of the rules $h_1$ and $h_2$ in terms of the empirical agreement rate between the two rules. This bound formally justifies both the use

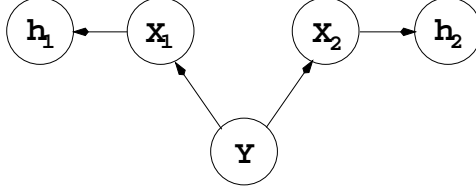

Figure 1: The co-training scenario with rules $h_1$ and $h_2$.

of agreement in the objective function and the use of partial rules. The bound shows the potential power of unlabeled data — low generalization error can be achieved for complex rules with a sufficient quantity of *unlabeled* data. The use of partial rules is analogous to the use of confidence ratings — a partial rule is just a rule with two levels of confidence. So the bound can also be viewed as justifying the partial labeling aspect of rule-based bootstrapping, at least in the case of co-training where an independence assumption holds. The generalization bound leads naturally to algorithms for optimizing the bound. A simple greedy procedure for doing this is quite similar to the co-training algorithm suggested by Collins and Singer.

## 2   The Main Result

We start with some basic definitions and observations. Let $S$ be an i.i.d. sample consisting of individual samples $s_1$, ..., $s_m$. For any statement $\Phi[s]$ we let $S(\Phi)$ be the subset $\{s_j : \Phi[s_j]\}$. For any two statements $\Phi$ and $\Psi$ we define the empirical estimate $\widehat{P}(\Phi \mid \Psi)$ to be $|S(\Phi \wedge \Psi)|/|S(\Psi)|$. For the co-training bounds proved here we assume data is drawn from some distribution over triples $\langle x_1, y, x_2 \rangle$ with $x_1 \in \mathcal{X}_1, y \in \{1, \ldots, k\}$, and $x_2 \in \mathcal{X}_2$, and where $x_1$ and $x_2$ are conditionally independent given $y$, that is, $P(x_1|y, x_2) = P(x_1|y)$ and $P(x_2|y, x_1) = P(x_2|y)$. In the co-training framework we are given an unlabeled sample $S_U$ of pairs $\langle x_1, x_2 \rangle$ drawn i.i.d. from the underlying distribution, and possibly some labeled samples $S_L$. We will mainly be interested in making inferences from the unlabeled data. A *partial rule* $h$ on a set $\mathcal{X}$ is a mapping from $\mathcal{X}$ to $\{1, \ldots, k, \bot\}$. We will be interested in pairs of partial rules $h_1$ and $h_2$ which largely agree on the unlabeled data.

The conditional probability relationships in our scenario are depicted graphically in figure 1. Important intuition is given by the *data-processing inequality* of information theory (Cover and Thomas, 1991): $I(h_1; y) \geq I(h_1; h_2)$. In other words, any mutual information between $h_1$ and $h_2$ must be mediated through $y$. In particular, if $h_1$ and $h_2$ agree to a large extent, then they must reveal a lot about $y$. And yet finding such a pair $(h_1, h_2)$ requires no labeled data at all. This simple observation is a major motivation for the proof, but things are complicated considerably by partial rules and by approximate agreement.

For a given partial rule $h_1$ with $P(h_1 \neq \bot) > 0$ define a function $f$ on $\{1, \ldots, k\}$ by

$$f(l) = arg \max_{1 \leq i \leq k} P(h_1 = i \mid y = l)$$

We want $h_1$ to be a nearly deterministic function of the actual label $y$; in other words, we want $P(h_1 = f(y) \mid h_1 \neq \bot)$ to be near one. We would also like $h_1$ to carry the same information as $y$. This is equivalent to saying that $f$ should be a permutation of the possible labels $\{1, \ldots, k\}$. Here we give a condition using only unlabeled data which guarantees, up to high confidence, that $f$ is a permutation; this is the best we can hope for using unlabeled

data alone. We also bound the error rates $P(h_1 \neq i \mid f(y) = i, \ h_1 \neq \bot)$ using only unlabeled data. In the case of $k = 2$, if $f$ is a permutation then $f$ is either the identity function or the function reversing the two possible values. We use the unlabeled data to select $h_1$ and $h_2$ so that $f$ is a permutation and $h_1$ has low error rate. We can then use a smaller amount of labeled data to determine which permutation we have found.

We now introduce a few definitions related to sampling issues. Some measure of the complexity of rules $h_1$ and $h_2$ is needed; rather than VC dimension, we adopt a clean notion of bit length. We assume that rules are specified in some rule language and write $|h|$ for the number of bits used to specify the rule $h$. We assume that the rule language is prefix-free (no proper prefix of the bit string specifying a rule is itself a legal rule specification). A prefix free code satisfies the Kraft inequality $\sum_h 2^{-|h|} \leq 1$. For given partial rules $h_1$ and $h_2$ and $i \in \{1, \ldots, k\}$ we now define the following functions of the sample $S$. The first, as we will see, is a bound on the sampling error for empirical probabilities conditioned upon $h_2 = i, h_1 \neq \bot$. The second is a sampling-adjusted disagreement rate between $h_1$ and $h_2$.

$$\epsilon_i(h_1, h_2, \delta) \quad = \quad \sqrt{\frac{(\ln 2)(|h_1| + |h_2|) + \ln \frac{2k}{\delta}}{2|S(h_2 = i, \ h_1 \neq \bot)|}}$$

$$\gamma_i(h_1, h_2, \delta) \quad = \quad \widehat{P}(h_1 = i \mid h_2 = i, h_1 \neq \bot) - \widehat{P}(h_1 \neq i \mid h_2 = i, h_1 \neq \bot) - 2\epsilon_i(h_1, h_2, \delta)$$

Note that if the sample size is sufficiently large (relative to $|h_1|$ and $|h_2|$) then $\epsilon_i(h_1, h_2, \delta)$ is near zero. Also note that if $h_1$ and $h_2$ have near perfect agreement when they both are not $\bot$ then $\gamma_i(h_1, h_2, \delta)$ is near one. We can now state our main result.

**Theorem 1** *With probability at least $1 - \delta$ over the choice of the sample $S$, we have that for all $h_1$ and $h_2$, if $\gamma_i(h_1, h_2, \delta) > 0$ for $1 \leq i \leq k$ then (a) $f$ is a permutation and (b) for all $1 \leq i \leq k$,*

$$P(h_1 \neq i \mid f(y) = i, h_1 \neq \bot) \quad \leq \quad \frac{\widehat{P}(h_1 \neq i \mid h_2 = i, h_1 \neq \bot) + \epsilon_i(h_1, h_2, \delta)}{\gamma_i(h_1, h_2, \delta)}.$$

The theorem states, in essence, that if the sample size is large, and $h_1$ and $h_2$ largely agree on the unlabeled data, then $\widehat{P}(h_1 \neq i \mid h_2 = i, h_1 \neq \bot)$ is a good estimate of the error rate $P(h_1 \neq i \mid f(y) = i, h_1 \neq \bot)$.

The theorem also justifies the use of partial rules. Of course it is possible to convert a partial rule to a total rule by forcing a random choice when the rule would otherwise return $\bot$. Converting a partial rule to a total rule in this way and then applying the above theorem to the total rule gives a weaker result. An interesting case is when $k = 2$, $h_2$ is total and is a perfect copy of $y$, and $P(h_1 \neq \bot)$ happens to be $1/\sqrt{|S|}$. In this case the empirical error rate of the corresponding total rule — the rule that guesses when $h_1$ has no opinion — will be statistically indistinguishable from from 1/2. However, in this case theorem 1 can still establish that the false positive and false negative rate of the *partial* rule $h_1$ is near zero.

## 3  The Analysis

We start with a general lemma about conditional probability estimation.

**Lemma 2** *For any i.i.d. sample $S$, and any statements $\Phi$ and $\Psi$ about individual instances in the sample, the following holds with probability at least $1 - \gamma$ over the choice of $S$.*

$$\left| P(\Phi \mid \Psi) - \widehat{P}(\Phi \mid \Psi) \right| \quad \leq \quad \sqrt{\frac{\ln 2/\gamma}{2|S(\Psi)|}} \tag{1}$$

*Proof.* We have the following where the third step follows by the Chernoff bound.

$$P\left(\left|P(\Phi \mid \Psi) - \widehat{P}(\Phi \mid \Psi)\right| > \sqrt{\tfrac{\ln 2/\gamma}{2|S(\Psi)|}}\right)$$

$$= \sum_k P(|S(\Psi)| = k) \cdot P\left(\left|P(\Phi \mid \Psi) - \widehat{P}(\Phi \mid \Psi)\right| > \sqrt{\tfrac{\ln 2/\gamma}{2|S(\Psi)|}} \ \middle| \ |S(\Psi)| = k\right)$$

$$\leq \sum_k P(|S(\Psi)| = k) \cdot \gamma \quad = \quad \gamma$$

∎

Therefore, with probability at least $1 - 2^{-|h_1|}2^{-|h_2|}\delta/k$,

$$|\widehat{P}(h_1 = i \mid h_2 = i, h_1 \neq \bot) - P(h_1 = i \mid h_2 = i, h_1 \neq \bot)| \leq \epsilon_i(h_1, h_2, \delta) \quad (2)$$

for any given $h_1, h_2$. By the union bound and the Kraft inequality, we have that with probability at least $1 - \delta$ this must hold simultaneously for all $h_1$ and $h_2$, and all $1 \leq i \leq k$.

**Lemma 3** *Pick any rules $h_1$ and $h_2$ for which equation (2) as well as $\gamma_i(h_1, h_2, \delta) > 0$ hold for all $1 \leq i \leq k$. Then $f$ is a permutation, and moreover, for any $i$,*

$$P(h_1 = i \mid f(y) = i, h_1 \neq \bot) \ > \ 1/2.$$

*Proof.* Pick any $i \in \{1, \ldots, k\}$. We need to show that there exists some $j$ such that $f(j) = i$. By the definition of $\gamma_i$ and condition (2) we know

$$P(h_1 = i \mid h_2 = i, h_1 \neq \bot) - P(h_1 \neq i \mid h_2 = i, h_1 \neq \bot) \geq \gamma_i(h_1, h_2, \delta)$$

Since $\gamma_i(h_1, h_2, \delta) > 0$, it follows that $P(h_1 = i \mid h_2 = i, h_1 \neq \bot) > 1/2$. Rewriting this by conditioning on $y$, we get

$$\sum_{1 \leq j \leq k} P(y = j \mid h_2 = i, h_1 \neq \bot)P(h_1 = i \mid y = j, \ h_1 \neq \bot) \ > \ 1/2.$$

The summation is a convex combination; therefore there must exist some $j$ such that $P(h_1 = i \mid y = j, h_1 \neq \bot) > 1/2$. So for each $i$ there must exist a $j$ with $f(j) = i$, whereby $f$ is a permutation. ∎

**Lemma 4** *Pick any rules $h_1$ and $h_2$ satisfying the conditions of the previous lemma. Then $P(f(y) = i \mid h_2 = i, h_1 \neq \bot)$ is at least $\gamma_i(h_1, h_2, \delta)$.*

*Proof.* By the previous lemma $f$ is a permutation, so $f(y)$ has the same information content as $y$. Therefore $x_1$ and $x_2$ are conditionally independent given $f(y)$. For any $i$,

$$\gamma_i(h_1, h_2, \delta)$$
$$\leq \ P(h_1 = i \mid h_2 = i, h_1 \neq \bot) - P(h_1 \neq i \mid h_2 = i, h_1 \neq \bot)$$
$$= \ \sum_j P(f(y) = j \mid h_2 = i, h_1 \neq \bot)\left(\begin{array}{l} P(h_1 = i \mid f(y) = j, h_1 \neq \bot) \\ -P(h_1 \neq i \mid f(y) = j, h_1 \neq \bot) \end{array}\right)$$
$$\leq \ P(f(y) = i \mid h_2 = i, h_1 \neq \bot)\left(\begin{array}{l} P(h_1 = i \mid f(y) = i, h_1 \neq \bot) \\ -P(h_1 \neq i \mid f(y) = i, h_1 \neq \bot) \end{array}\right)$$
$$+ \sum_{j \neq i} P(f(y) = j \mid h_2 = i, h_1 \neq \bot)\left(\begin{array}{l} P(h_1 = i \mid f(y) = j, h_1 \neq \bot) \\ -P(h_1 = j \mid f(y) = j, h_1 \neq \bot) \end{array}\right)$$

where the second step involves conditioning on $f(y)$. Also by the previous lemma, we have $P(h_1 = j \mid f(y) = j, h_1 \neq \perp) > 1/2$ so the second term in the above sum must be negative, whereby

$$
\begin{aligned}
\gamma_i(h_1, h_2, \delta) &\leq P(f(y) = i \mid h_2 = i, h_1 \neq \perp)\left(\begin{array}{c} P(h_1 = i \mid f(y) = i, h_1 \neq \perp) \\ -P(h_1 \neq i \mid f(y) = i, h_1 \neq \perp) \end{array}\right) \\
&\leq P(f(y) = i \mid h_2 = i, h_1 \neq \perp)
\end{aligned}
$$

∎

Under the conditions of these lemmas, we can derive the bounds on error rates:

$$
\begin{aligned}
P(h_1 \neq i \mid f(y) = i, h_1 \neq \perp) &\leq \frac{P(h_1 \neq i \mid h_2 = i, h_1 \neq \perp)}{P(f(y) = i \mid h_2 = i, h_1 \neq \perp)} \\
&\leq \frac{\widehat{P}(h_1 \neq i \mid h_2 = i, h_1 \neq \perp) + \epsilon_i(h_1, h_2, \delta)}{\gamma_i(h_1, h_2, \delta)}
\end{aligned}
$$

## 4    Bounding Total Error

Assuming that we make a random guess when $h_1 = \perp$, the total error rate of $h_1$ can be written as follows.

$$
\text{error}(h_1) = P(h_1 \neq \perp)P(h_1 \neq f(y) \mid h_1 \neq \perp) + \frac{k-1}{k}P(h_1 = \perp)
$$

To give a bound on the total error rate we first define $b_i(h_1, h_2, \delta)$ to be the bounds on the error rate for label $i$ given in theorem 1.

$$
b_i(h_1, h_2, \delta) = \frac{1}{\gamma_i(h_1, h_2, \delta)}\left(\widehat{P}(h_1 \neq i \mid h_2 = i, h_1 \neq \perp) + \epsilon_i(h_1, h_2, \delta)\right)
$$

We can now state a bound on the total error rate as a corollary to theorem 1.

**Corollary 5** *With probability at least $1 - \delta$ over the choice of $S$ we have the following for all pairs of rules $h_1$ and $h_2$ such that for all $i$ we have $\gamma_i(h_1, h_2, \delta/2) > 0$ and $b_i(h_1, h_2, \delta/2) \leq (k-1)/k$.*

$$
\begin{aligned}
\text{error}(h_1) &\leq \left(\widehat{P}(h_1 \neq \perp) - \epsilon(|h_1|, \delta/2)\right)\max_j b_j(h_1, h_2, \delta/2) \\
&\quad + \frac{k-1}{k}\left(\widehat{P}(h_1 = \perp) + \epsilon(|h_1|, \delta/2)\right) \\
\epsilon(k, \delta) &= \sqrt{\frac{k \ln 2 + \ln 2/\delta}{2|S|}}
\end{aligned}
$$

*Proof.* From our main theorem, we know that with probability at least $1 - \delta/2$, for all $i$. $P(h_1 \neq i \mid f(y) = i, h_1 \neq \perp)$ is bounded by $b_i(h_1, h_2, \delta/2)$. This implies that with probability at least $1 - \delta/2$,

$$
\text{error}(h_1) \leq P(h_1 \neq \perp)\max_j b_j(h_1, h_2, \delta/2) + \frac{k-1}{k}(1 - P(h_1 \neq \perp)) \tag{3}
$$

With probability at least $1 - \delta/2$ we have that $|\widehat{P}(h_1 \neq \bot) - P(h_1 \neq \bot)|$ is no larger than $\epsilon(|h_1|, \delta/2)$. So by the union bound both of these conditions hold simultaneously with probability at least $1 - \delta$. Since $\max_j b_j(h_1, h_2, \delta/2) \leq (k-1)/k$ we have that the upper bound in (3) is maximized by setting $P(h_1 \neq \bot)$ equal to $\widehat{P}(h_1 \neq \bot) - \epsilon(|h_1|, \delta/2)$. ∎

Corollary 5 can be improved in a variety of ways. One could use a relative Chernoff bound to tighten the uncertainty in $P(h_1 \neq \bot)$ in the case where this probability is small. One could also use the error rate bounds $b_i(h_1, h_2, \delta/2)$ to construct bounds on $P(f(y) = i \mid h_1 \neq \bot)$. One could then replace the max over $b_i(h_1, h_2, \delta/2)$ by a convex combination. Another approach is to use the error rate of a rule that combines $h_1$ and $h_2$, e.g., the rule outputs $h_1$ if $h_1 \neq \bot$, otherwise outputs $h_2$ if $h_2 \neq \bot$, and otherwise guesses a random value. This combined rule will have a lower error rate and it is possible to give bounds on the error rate of the combined rule. We will not pursue these refined bounds here. It should be noted, however, that the algorithm described in section 4 can be used with any bound on total error rate.

## 5 A Decision List Algorithm

This section suggests a learning algorithm inspired by both Yarowsky (1995) and Collins and Singer (1999) but modified to take into account theorem 1 and Corollary 5. Corollary 5, or some more refined bound on total error, provides an objective function that can be pursued by a learning algorithm — the objective is to find $h_1$ and $h_2$ so as to minimize the upper bound on the total error rate. Typically, however, the search space is enormous. Following Yarowsky, we consider the greedy construction of decision list rules.

Let $\mathcal{F}_1$ and $\mathcal{F}_2$ be two "feature sets" such that for $p \in \mathcal{F}_1$ and $x_1 \in \mathcal{X}_1$ we have $p(x_1) \in \{0, 1\}$ and for $q \in \mathcal{F}_2$ and $x_2 \in \mathcal{X}_2$ we have $q(x_2) \in \{0, 1\}$. We assume that $h_1$ is to be a decision list over the features in $\mathcal{F}_1$, i.e., a finite sequence of the form $\langle p_1, \ v_1 \rangle; \ldots; \langle p_k, \ v_k \rangle$ where $p_i \in \mathcal{F}_1$ and $v_i \in \{1, \ldots, k\}$. A decision list can be viewed as a right-branching decision tree. More specifically, if $h_1$ is the list $\langle p_1, \ v_1 \rangle; \langle p_2, v_2 \rangle; \ldots; \langle p_k, \ v_k \rangle$ then $h_1(x_1)$ is $v_1$ if $p_1(x_1) = 1$ and otherwise equals the value of the list $\langle p_2, v_2 \rangle; \ldots; \langle p_k, \ v_k \rangle$ on $x_1$. We define an empty decision list to have value $\bot$. For $\epsilon$ in $(0, 1)$ we can define $|h_1|$ as follows where $k$ is the number of feature-value pairs in $h_1$.

$$|h_1| = \log_2 \frac{1}{\epsilon} + k \log_2 \frac{k|\mathcal{F}_1|}{1 - \epsilon}$$

It is possible to show that $2^{-|h_1|}$ equals the probability that a certain stochastic process generates the rule $h_1$. This implies the Kraft inequality $\sum_h 2^{-|h|} \leq 1$ which is all that is needed in theorem 1 and corollary 5. We also assume that $h_2$ is a decision list over the features $\mathcal{F}_2$ and define $|h_2|$ similarly.

Following Yarowsky we suggest growing the decision lists in a greedy manner adding one feature value pair at a time. A natural choice of greedy heuristic might be a bound on the total error rate. However, in many cases the final objective function is not an appropriate choice for the greedy heuristic in greedy algorithms. A* search, for example, might be viewed as a greedy heuristic where the heuristic function estimates the number of steps needed to reach a high-value configuration — a low value configuration might be one step away from a high value configuration. The greedy heuristic used in greedy search should estimate the value of the final configuration. Here we suggest using $\max_j b_j(h_1, h_2, \delta/2)$ as a heuristic estimate of the final total error rate — in the final configuration we should

have that $P(h_1 \neq \perp)$ is reasonably large and the important term will be $\max_j b_j(h_1, h_2, \delta)$. For concreteness, we propose the following algorithm. Many variants of this algorithm also seem sensible.

1. Initialize $h_1$ and $h_2$ to "seed rule" decision lists using domain-specific prior knowledge.

2. Until $\widehat{P}(h_1 = \perp)$ and $\widehat{P}(h_2 = \perp)$ are both zero, or all features have been used in both rules, do the following.

   (a) Let $g$ denote $h_1$ if $\widehat{P}(h_1 = \perp) > \widehat{P}(h_2 = \perp)$ and $h_2$ otherwise.
   (b) If $\gamma_i(h_1, h_2, \delta/2) \leq 0$ for some $i$, then extend $g$ by the pair $\langle p, v \rangle$ which most increases $\min_i \gamma_i(h_1, h_2, \delta/2)$.
   (c) Otherwise extend $g$ by a single feature-value pair selected to minimize the $\max_j b_j(h_1, h_2, \delta/2)$.

3. Prune the rules — iteratively remove the pair from the end of either $h_1$ or $h_2$ that greedily minimizes the bound on total error until no such removal reduces the bound.

## 6   Future Directions

We have given some theoretical justification for some aspects of co-training algorithms that have been shown to work well in practice. The co-training assumption we have used in our theorems are is at best only approximately true in practice. One direction for future research is to try to relax this assumption somehow. The co-training assumption states that $x_1$ and $x_2$ are independent given $y$. This is equivalent to the statement that the mutual information between $x_1$ and $x_2$ given $y$ is zero. We could relax this assumption by allowing some small amount of mutual information between $x_1$ and $x_2$ given $y$ and giving bounds on error rates that involve this quantity of mutual information. Another direction for future work, of course, is the empirical evaluation of co-training and bootstrapping methods suggested by our theory.

**Acknowledgments**

The authors wish to acknowledge Avrim Blum for useful discussions and give special thanks to Steve Abney for clarifying insights.

**Literature cited**

Blum, A. & Mitchell, T. (1998) Combining labeled and unlabeled data with co-training. *COLT*.
Collins, M. & Singer, Y. (1999) Unsupervised models for named entity classification. *EMNLP*.
Cover, T. & Thomas, J. (1991) *Elements of information theory*. Wiley.
Dempster, A., Laird, N. & Rubin, D. (1977) Maximum-likelihood from incomplete data via the EM algorithm. *J. Royal Statist. Soc. Ser. B*, 39:1-38.
Nigam, K. & Ghani, R. (2000) Analyzing the effectiveness and applicability of co-training. *CIKM*.
Yarowsky, D. (1995) Unsupervised word sense disambiguation rivaling supervised methods. *ACL*.
